# Analyzing Coupled Brain Sources: Distinguishing True from Spurious Interaction

**Guido Nolte**[1]**, Andreas Ziehe**[3]**, Frank Meinecke**[1] **and Klaus-Robert Müller**[1,2]

[1] Fraunhofer FIRST.IDA, Kekuléstr. 7, 12489 Berlin, Germany
[2] Dept. of CS, University of Potsdam, August-Bebel-Strasse 89, 14482 Potsdam, Germany
[3] TU Berlin, Inst. for Software Engineering, Franklinstr. 28/29, 10587 Berlin, Germany
{nolte,ziehe,meinecke,klaus}@first.fhg.de

## Abstract

When trying to understand the brain, it is of fundamental importance to analyse (e.g. from EEG/MEG measurements) what parts of the cortex interact with each other in order to infer more accurate models of brain activity. Common techniques like Blind Source Separation (BSS) can estimate brain sources and single out artifacts by using the underlying assumption of source signal independence. However, physiologically interesting brain sources typically interact, so BSS will—by construction—fail to characterize them properly. Noting that there are truly interacting sources and signals that only seemingly interact due to effects of volume conduction, this work aims to contribute by *distinguishing* these effects. For this a new BSS technique is proposed that uses anti-symmetrized cross-correlation matrices and subsequent diagonalization. The resulting decomposition consists of the truly interacting brain sources and suppresses any spurious interaction stemming from volume conduction. Our new concept of interacting source analysis (ISA) is successfully demonstrated on MEG data.

## 1 Introduction

Interaction between brain sources, phase synchrony or coherent states of brain activity are believed to be fundamental for neural information processing (e.g. [2, 6, 5]). So it is an important topic to devise new methods that can more reliably characterize interacting sources in the brain. The macroscopic nature and the high temporal resolution of electroencephalography (EEG) and magnetoencephalography (MEG) in the millisecond range makes these measurement technologies ideal candidates to study brain interactions. However, interpreting data from EEG/MEG channels in terms of connections between brain sources is largely hampered by artifacts of volume conduction, i.e. the fact that activities of single sources are observable as superposition in all channels (with varying amplitude). So ideally one would like to discard all—due to volume conduction—seemingly interacting signals and retain only truly linked brain source activity.

So far neither existing source separation methods nor typical phase synchronization anal-

ysis (e.g. [1, 5] and references therein) can adequately handle signals when the sources are both superimposed and interacting i.e. non-independent (cf. discussions in [3, 4]). It is here where we contribute in this paper by proposing a new algorithm to distinguish true from spurious interaction. A prerequisite to achieve this goal was recently established by [4]: as a consequence of instantaneous and linear volume conduction, the cross-spectra of independent sources are real-valued, regardless of the specifics of the volume conductor, number of sources or source configuration. Hence, a non-vanishing imaginary part of the cross-spectra must necessarily reflect a true interaction. Drawbacks of Nolte's method are: (a) cross-spectra for all frequencies in multi-channel systems contain a huge amount of information and it can be tedious to find the interesting structures, (b) it is very much possible that the interacting brain consists of several subsystems which are independent of each other but are not separated by that method, and (c) the method is well suited for rhythmic interactions while wide-band interactions are not well represented.

A recent different approach by [3] uses BSS as preprocessing step before phase synchronization is measured. The drawback of this method is the assumption that there are not more sources than sensors, which is often heavily violated because, e.g., channel noise trivially consists of as many sources as channels, and, furthermore, brain noise can be very well modelled by assuming thousands of randomly distributed and independent dipoles.

To avoid the drawbacks of either method we will formulate an algorithm called interacting source analysis (ISA) which is technically based on BSS using second order statistics but is only sensitive to interacting sources and, thus, can be applied to systems with *arbitrary* noise structure. In the next section, after giving a short introduction to BSS as used for this paper, we will derive some fundamental properties of our new method. In section 3 we will show in simulated data and real MEG examples that the ISA procedure finds the interacting components and separates interacting subsystems which are independent of each other.

## 2   Theory

The fundamental assumption of ICA is that a data matrix $X$, without loss of generality assumed to be zero mean, originates from a superposition of independent sources $S$ such that

$$X = AS \tag{1}$$

where $A$ is called the mixing matrix which is assumed to be invertible. The task is to find $A$ and hence $S$ (apart from meaningless ordering and scale transformations of the columns of $A$ and the rows of $S$) by merely exploiting statistical independence of the sources. Since independence implies that the sources are uncorrelated we may choose $W$, the estimated inverse mixing matrix, such that the covariance matrix of

$$\hat{S} \equiv WX \tag{2}$$

is equal to the identity matrix. This, however, does not uniquely determine $W$ because for any such $W$ also $UW$, where $U$ is an arbitrary orthogonal matrix, leads to a unit covariance matrix of $\hat{S}$. Uniqueness can be restored if we require that $W$ not only diagonalizes the covariance matrix but also cross-correlation matrices for various delays $\tau$, i.e. we require that

$$WC^X(\tau)W^\dagger = diag \tag{3}$$

with

$$C^X(\tau) \equiv \langle \mathbf{x}(t)\mathbf{x}^\dagger(t+\tau) \rangle \tag{4}$$

where $\mathbf{x}(t)$ is the $t.th$ column of $X$ and $\langle . \rangle$ means expectation value which is estimated by the average over $t$. Although at this stage all expressions are real-valued we introduce a complex formulation for later use.

Note, that since under the ICA assumption the cross-correlation matrices $C^S(\tau)$ of the source signals are diagonal

$$C^S_{ij}(\tau) = \langle s_i(t)s_i(t+\tau)\rangle\delta_{ij} = C^S_{ji}, \tag{5}$$

the cross-correlation matrices of the mixtures are symmetric:

$$C^X(\tau) = AC^S(\tau)A^\dagger = \left(AC^S(\tau)A^\dagger\right)^\dagger = C^{X\dagger}(\tau) \tag{6}$$

Hence, the antisymmetric part of $C^X(\tau)$ can only arise due to meaningless fluctuations and can be ignored. In fact, the above TDSEP algorithm uses symmetrized versions of $C^X(\tau)$ [8].

Now, the key and new point of our method is that we will turn the above argument upside down. Since non-interacting sources do not contribute (systematically) to the anti-symmetrized correlation matrices

$$D(\tau) \equiv C^X(\tau) - C^{X\dagger}(\tau) \tag{7}$$

any (significant) non-vanishing elements in $D(\tau)$ must arise from interacting sources, and hence the analysis of $D(\tau)$ is ideally suited to study the interacting brain. In doing so we exploit that neuronal interactions necessarily take some time which is well above the typical time resolution of EEG/MEG measurements.

It is now our goal to identify one or many interacting systems from a suitable spatial transformation which corresponds to a demixing of the systems rather than individual sources. Although we concentrate on those components which explicitly violate the independence assumption we will use the technique of simultaneous diagonalization to achieve this goal. We first note that a diagonalization of $D(\tau)$ using a real-valued $W$ is meaningless since with $D(\tau)$ also $WD(\tau)W^\dagger$ is anti-symmetric and always has vanishing diagonal elements. Hence $D(\tau)$ can only be diagonalized with a complex-valued $W$ with subsequent interpretation of it in terms of a real-valued transformation.

We will here discuss the case where all interacting systems consist of pairs of neuronal sources. Properties of systems with more than two interacting systems will be discussed below. Furthermore, for simplicity we assume an even number of channels. Then a real-valued spatial transformation $W_1$ exists such that the set of $D(\tau)$ becomes decomposed into $K = N/2$ blocks of size $2 \times 2$

$$W_1D(\tau)W_1^\dagger = \begin{pmatrix} \alpha_1(\tau)\begin{pmatrix} 0 & 1 \\ -1 & 0 \end{pmatrix} & 0 & 0 \\ 0 & \ddots & 0 \\ 0 & 0 & \alpha_K(\tau)\begin{pmatrix} 0 & 1 \\ -1 & 0 \end{pmatrix} \end{pmatrix} \tag{8}$$

Each block can be diagonalized e.g. with

$$\tilde{W}_2 = \begin{pmatrix} 1 & -i \\ 1 & i \end{pmatrix} \tag{9}$$

and with

$$W_2 = id_{K \times K} \otimes \tilde{W}_2 \tag{10}$$

we get

$$W_2W_1D(\tau)W_1^\dagger W_2^\dagger = diag \tag{11}$$

From a simultaneous diagonalization of $D(\tau)$ we obtain an estimate of the demixing matrix $\hat{W}$ of the true demixing matrix $W = W_2W_1$. We are interested in the columns of $W_1^{-1}$ which correspond to the spatial patterns of the interacting sources. Let us denote the $N \times 2$

submatrix of a matrix $B$ consisting of the $(2k-1).th$ and the $2k.th$ column as $(B)_k$. Then we can write

$$(W_1^{-1})_k \sim (W^{-1})_k \tilde{W}_2 \qquad (12)$$

and hence the desired spatial patterns of the $k.th$ system are a complex linear superposition of the $(2k-1).th$ and the $2k.th$ column of $W$. The subspace spanned in channel-space by the two interacting sources, denoted as $span((A)_k)$, can now be found by separating real and imaginary part of $W^{-1}$

$$span((A)_k) = span\left(\left(\Re((W^{-1})_k), \Im((W^{-1})_k)\right)\right) \qquad (13)$$

According to (13) we can calculate from $W$ just the 2D-subspaces spanned by the interacting systems but not the patterns of the sources themselves. The latter would indeed be impossible because all we analyze are anti-symmetric matrices which are, for each system, constructed as anti-symmetric outer products of the two respective field patterns. These anti-symmetric matrices are, apart from an irrelevant global scale, invariant with respect to a linear and real-valued mixing of the sources within each system.

The general procedure can now be outlined as follows.

1. From the data construct anti-symmetric cross-correlation matrices as defined in Eq.(7) for reasonable set of delays $\tau$.

2. Find a complex matrix $W$ such that $WD(\tau)W^{\dagger}$ is approximately diagonal for all $\tau$.

3. If the system consists of subsystems of paired interactions (and indeed, according to our own experience, very much in practice) the diagonal elements in $WD(\tau)W^{\dagger}$ come in pairs in the form $\pm i\lambda$. Each pair constitutes one interacting system. The corresponding two columns in $W^{-1}$, with separated real and imaginary parts, form an $N \times 4$ matrix $V$ with rank 2. The span of $V$ coincides with the space spanned by the respective system. In practice, $V$ will have two singular values which are just very small rather than exactly zero. The corresponding singular vectors should then be discarded. Instead of analyzing $V$ in the above way it is also possible to simply take the real and imaginary part of either one of the two columns.

4. Similar to the spatial analysis, it is not possible to separate the time-courses of two interacting sources within one subsystem. In general, two *estimated* time-courses, say $\hat{s}_1(t)$ and $\hat{s}_2(t)$, are an unknown linear combination of the true source activations $s_1(t)$ and $s_2(t)$. To understand the type of interaction it is still recommended to look at the power and autocorrelation functions. Invariant with respect to linear mixing with one subsystem is the anti-symmetrized cross-correlation between $\hat{s}_1(t)$ and $\hat{s}_2(t)$ and, equivalently, the imaginary part of the cross-spectral density. For the $k.th$ system, these quantities are given by the $k.th$ diagonal $\lambda_k(\tau)$ and their respective Fourier transforms.

While (approximate) simultaneous diagonalization of $D(\tau)$ using complex demixing matrices is always possible with pairwise interactions we can expect only block-diagonal structure if a larger number of sources are interacting within one or more subsystems. We will show below for simulated data that the algorithm still finds these blocks although the actual goal, i.e. diagonal $WD(\tau)W^{\dagger}$, is not reachable.

# 3 Results

## 3.1 Simulated data

Matrices were approximately simultaneously diagonalized with the DOMUNG-algorithm
[7], which was generalized to the complex domain. Here, an initial guess for the demixing
matrix $W$ is successively optimized using a natural gradient approach combined with line
search according to the requirement that the off-diagonals are minimal under the constraint
$det(W) = 1$. Special care has to be taken in the choice of the initial guess. Due to the
complex-conjugation symmetry of our problem (i.e., $W^*$ diagonalizes as well as $W$) the
initial guess may not be set to a real-valued matrix because then the component of the
gradient in imaginary direction will be zero and $W$ will converge to a real-valued saddle
point.

We simulated two random interacting subsystems of dimensions $N_A$ and $N_B$ which were
assumed to be mutually independent. The two subsystems were mapped into $N =
N_A + N_B$ channels with a random mixture matrix. The anti-symmetrized cross-correlation
matrices read

$$D(\tau) = A \begin{pmatrix} D_A(\tau) & 0 \\ 0 & D_B(\tau) \end{pmatrix} A^\dagger \tag{14}$$

where $A$ is a random real-valued $N \times N$ matrix, and $D_A(\tau)$ ($D_B(\tau)$), with $\tau = 1...20$, are
a set of random anti-symmetric $N_A \times N_A$ ($N_B \times N_B$) matrices. Note, that in this context,
$\tau$ has no physical meaning.

As expected, we have found that if one of the subsystems is two-dimensional the respec-
tive block can always be diagonalized exactly for any number of $\tau$s. We have also seen,
that the diagonalization procedure always perfectly separates the two subsystems even if
a diagonalization within a subsystem is not possible. A typical result for $N_A = 2$ and
$N_B = 3$ is presented in Fig.1. In the left panel we show the average of the absolute value
of correlation matrices before spatial mixing. In the middle panel we show the respective
result after random spatial mixture and subsequent demixing, and in the right panel we
show $W_1 A$ where $W_1$ is the estimated real version of the demixing matrix as explained in
the preceding section. We note again, that also for the two-dimensional block, which can
always be diagonalized exactly, one can only recover the corresponding two-dimensional
subspace and not the source components themselves.

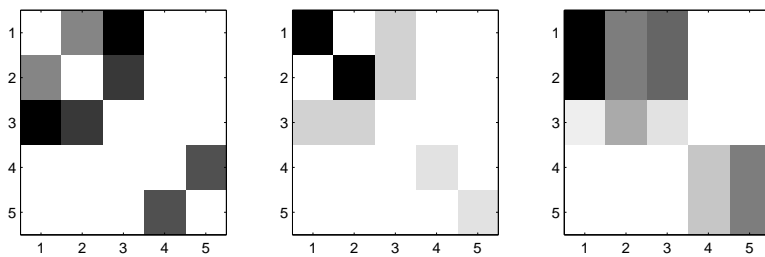

Figure 1: Left: average of the absolute values of correlation matrices before spatial mixing;
middle: same after random spatial mixture and subsequent demixing; right: product of the
estimated demixing matrix and the true mixing matrix ($W_1 A$). White indicates zero and
black the maximum value for each matrix.

## 3.2 Real MEG data

We applied our method to real data gathered in 93 MEG channels during triggered finger movements of the right or left hand. We recall that for each interacting component we get two results: a) the 2D subspace spanned by the two components and b) the diagonals of the demixed system, say $\pm i\lambda_k(\tau)$. To visualize the 2D subspace in a unique way we construct from the two patterns of the $k.th$ system, say $\mathbf{x}_1$ and $\mathbf{x}_2$, the anti-symmetric outer product

$$D_k \equiv \mathbf{x}_1\mathbf{x}_2^T - \mathbf{x}_2\mathbf{x}_1^T \tag{15}$$

Indeed, the $k.th$ subsystem contributes this matrix to the anti-symmetrized cross-correlations $D(\tau)$ with varying amplitude for all $\tau$.

The matrix $D_k$ is now visualized as shown in Figs.3. The $i.th$ row of $D_k$ corresponds to the interaction of the $i.th$ channel to all others and this interaction is represented by the contour-plot within the $i.th$ circle located at the respective channel location. In this example, the observed structure clearly corresponds to the interaction between eye-blinks and visual cortex since occipital channels interact with channels close to the eyes and vice versa.

In the upper panels of Fig.2 we show the corresponding temporal and spectral structures of this interaction, represented by $\lambda_k(\tau)$, and its Fourier transform, respectively. We observe in the temporal domain a peak at a delay around 120 ms (indicated by the arrow) which corresponds well to the response time of the primary visual cortex to visual input.

In the lower panels of Fig.2 we show the temporal and spectral pattern of another interacting component with a clear peak in the alpha range (10 Hz). The corresponding spatial pattern (Fig.4) clearly indicates an interacting system in occipital-parietal areas.

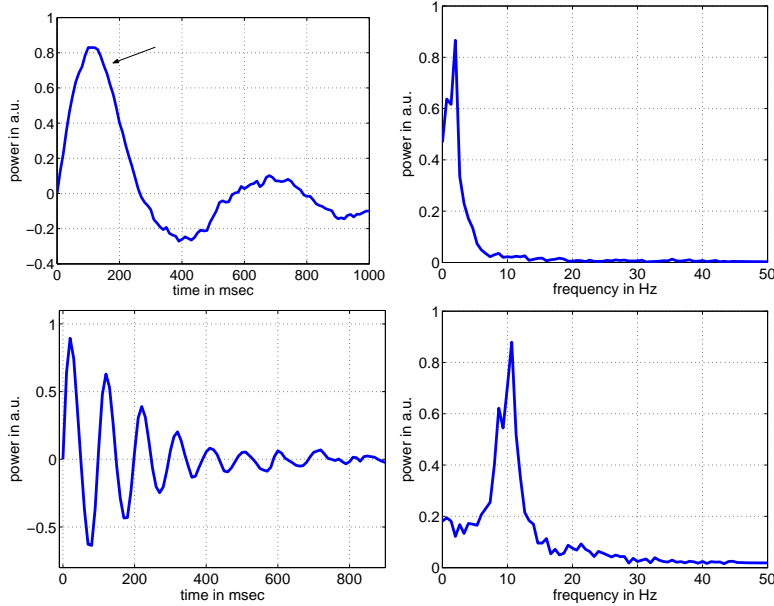

Figure 2: Diagonals of demixed antisymmetric correlation matrices as a function of delay $\tau$ (left panels) and, after Fourier transformation, as a function of frequency (right panels). Top: interaction of eye-blinks and visual cortex; bottom: interaction of alpha generators.

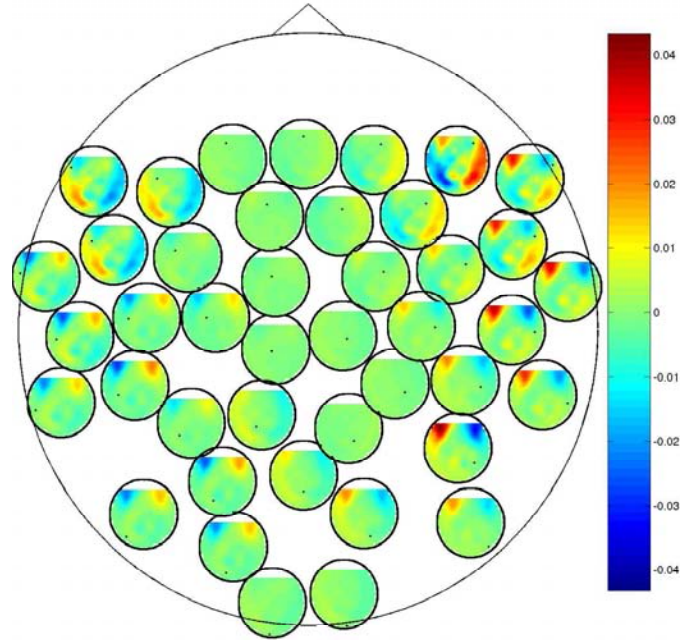

Figure 3: Spatial pattern corresponding to the interaction between eye-blinks and visual cortex.

## 4   Conclusion

When analyzing interaction between brain sources from macroscopic measurements like EEG/MEG it is important to distinguish physiologically reasonable patterns of interaction and spurious ones. In particular, volume conduction effects make large parts of the cortex seemingly interact although in reality such contributions are purely artifactual. Existing BSS methods that have been used with success for artifact removal and for estimation of brain sources will by construction fail when attempting to separate interacting i.e. non-independent brain sources. In this work we have proposed a new BSS algorithm that uses anti-symmetrized cross-correlation matrices and subsequent diagonalization and can thus reliably extract meaningful interaction while ignoring all spurious effects. Experiments using our interacting source analysis (ISA) reveal interesting relationships that are found *blindly*, e.g. inferring a component that links both eyes with visual cortex activity in a self-paced finger movement experiment. A more detailed look at the spectrum exhibits a peak at the typing frequency, and, in fact going back to the original MEG traces, eye-blinks were strongly coupled with the typing speed. This simple finding exemplifies that ISA is a powerful new technique for analyzing dynamical correlations in macroscopic brain measurements.

Future studies will therefore apply ISA to other neurophysiological paradigms in order to gain insights into the coherence and synchronicity patterns of cortical dynamics. It is especially of high interest to explore the possibilities of using true brain interactions as revealed by the imaginary part of cross-spectra as complementing information to improve the performance of brain computer interfaces.

**Acknowledgements.** We thank G. Curio for valuable discussions. This work was supported in part by the IST Programme of the European Community, under PASCAL Network

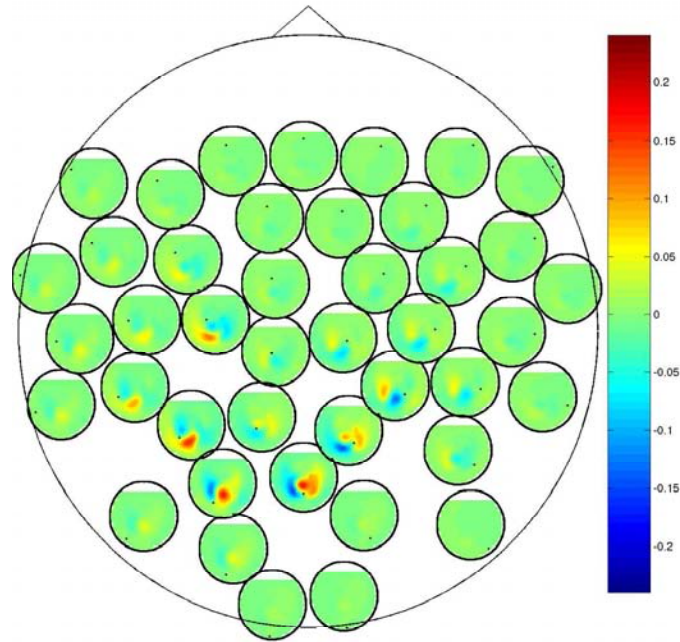

Figure 4: Spatial pattern corresponding to the interaction between alpha generators.

of Excellence, IST-2002-506778 and the BMBF in the BCI III project (grant 01BE01A). This publication only reflects the author's views.

## References

[1] A. Hyvarinen, J. Karhunen, and E. Oja. *Independent Component Analysis*. Wiley, 2001.

[2] V.K. Jirsa. Connectivity and dynamics of neural information processing. *Neuroinformatics*, (2):183–204, 2004.

[3] Frank Meinecke, Andreas Ziehe, Jürgen Kurths, and Klaus-Robert Müller. Measuring Phase Synchronization of Superimposed Signals. *Physical Review Letters*, 94(8), 2005.

[4] G. Nolte, O. Bai, L. Wheaton, Z. Mari, S. Vorbach, and M. Hallet. Identifying true brain interaction from eeg data using the imaginary part of coherency. *Clinical Neurophysiology*, 115:2292–2307, 2004.

[5] A. Pikovsky, M. Rosenblum, and J. Kurths. *Synchronization – A Universal Concept in Nonlinear Sciences*. Cambridge University Press, 2001.

[6] W. Singer. Striving for coherence. *Nature*, 397(6718):391–393, Feb 1999.

[7] A. Yeredor, A. Ziehe, and K.-R. Müller. Approximate joint diagonalization using a natural-gradient approach. In Carlos G. Puntonet and Alberto Prieto, editors, *Lecture Notes in Computer Science*, volume 3195, pages 89–96, Granada, 2004. Springer-Verlag. Proc. ICA 2004.

[8] A. Ziehe and K.-R. Müller. TDSEP – an efficient algorithm for blind separation using time structure. In L. Niklasson, M. Bodén, and T. Ziemke, editors, *Proceedings of the 8th International Conference on Artificial Neural Networks, ICANN'98*, Perspectives in Neural Computing, pages 675 – 680, Berlin, 1998. Springer Verlag.
